# Multi-scale Hyper-time Hardware Emulation of Human Motor Nervous System Based on Spiking Neurons using FPGA

**C. Minos Niu**
Department of Biomedical Engineering
University of Southern California
Los Angeles, CA 90089
minos.niu@sangerlab.net

**Sirish K. Nandyala**
Department of Biomedical Engineering
University of Southern California
Los Angeles, CA 90089
nandyala@usc.edu

**Won Joon Sohn**
Department of Biomedical Engineering
University of Southern California
Los Angeles, CA 90089
wonjsohn@gmail.com

**Terence D. Sanger**
Department of Biomedical Engineering
Department of Neurology
Department of Biokinesiology
University of Southern California
Los Angeles, CA 90089
terry@sangerlab.net

## Abstract

Our central goal is to quantify the long-term progression of pediatric neurological diseases, such as a typical 10-15 years progression of child dystonia. To this purpose, quantitative models are convincing only if they can provide multi-scale details ranging from neuron spikes to limb biomechanics. The models also need to be evaluated in hyper-time, i.e. significantly faster than real-time, for producing useful predictions. We designed a platform with digital VLSI hardware for multi-scale hyper-time emulations of human motor nervous systems. The platform is constructed on a scalable, distributed array of Field Programmable Gate Array (FPGA) devices. All devices operate asynchronously with 1 millisecond time granularity, and the overall system is accelerated to 365x real-time. Each physiological component is implemented using models from well documented studies and can be flexibly modified. Thus the validity of emulation can be easily advised by neurophysiologists and clinicians. For maximizing the speed of emulation, all calculations are implemented in combinational logic instead of clocked iterative circuits. This paper presents the methodology of building FPGA modules emulating a monosynaptic spinal loop. Emulated activities are qualitatively similar to real human data. Also discussed is the rationale of approximating neural circuitry by organizing neurons with sparse interconnections. In conclusion, our platform allows emulating pathological abnormalities such that motor symptoms will emerge and can be analyzed. It compels us to test the origins of childhood motor disorders and predict their long-term progressions.

## 1 Challenges of studying developmental motor disorders

There is currently no quantitative model of how a neuropathological condition, which mainly affects the function of neurons, ends up causing the functional abnormalities identified in clinical examinations. The gap in knowledge is particularly evident for disorders in developing human nervous systems, i.e. childhood neurological diseases. In these cases, the ultimate clinical effect of cellu-

lar injury is compounded by a complex interplay among the child's injury, development, behavior, experience, plasticity, etc. Qualitative insight has been provided by clinical experiences into the association between particular types of injury and particular types of outcome. Their quantitative linkages, nevertheless, have yet to be created – neither in clinic nor in cellular physiological tests. This discrepancy is significantly more prominent for individual child patients, which makes it very difficult to estimate the efficacy of treatment plans. In order to understand the consequence of injury and discover new treatments, it is necessary to create a modeling toolset with certain design guidelines, such that child neurological diseases can be quantitatively analyzed.

Perhaps more than any other organ, the brain necessarily operates on multiple spatial and temporal scales. On the one hand, it is the neurons that perform fundamental computations, but neurons have to interact with large-scale organs (ears, eyes, skeletal muscles, etc.) to achieve global functions. This multi-scale nature worths more attention in injuries, where the overall deficits depend on both the cellular effects of injuries and the propagated consequences. On the other hand, neural processes in developmental diseases usually operate on drastically different time scales, e.g. spinal reflex in milliseconds versus learning in years. Thus when studying motor nervous systems, mathematical modeling is convincing only if it can provide multi-scale details, ranging from neuron spikes to limb biomechanics; also the models should be evaluated with time granularity as small as 1 millisecond, meanwhile the evaluation needs to continue trillions of cycles in order to cover years of life.

It is particularly challenging to describe the multi-scale nature of human nervous system when modeling childhood movement disorders. Note that for a child who suffered brain injury at birth, the full development of all motor symptoms may easily take more than 10 years. Therefore the millisecond-based model needs to be evaluated significantly faster than real-time, otherwise the model will fail to produce any useful predictions in time. We have implemented realistic models for spiking motoneurons, sensory neurons, neural circuitry, muscle fibers and proprioceptors using VLSI and programmable logic technologies. All models are computed in Field Programmable Gate Array (FPGA) hardware in 365 times real-time. Therefore one year of disease progression can be assessed after one day of emulation. This paper presents the methodology of building the emulation platform. The results demonstrate that our platform is capable of producing physiologically realistic multi-scale signals, which are usually scarce in experiments. Successful emulations enabled by this platform will be used to verify theories of neuropathology. New treatment mechanisms and drug effects can also be emulated before animal experiments or clinical trials.

## 2  Methodology of multi-scale neural emulation

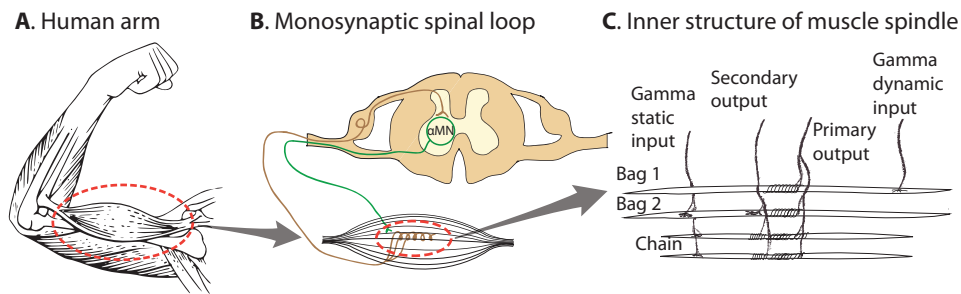

Figure 1: Illustration of the multi-scale nature of motor nervous system.

The motor part of human nervous system is responsible for maintaining body postures and generating voluntary movements. The multi-scale nature of motor nervous system is demonstrated in Fig.1. When the elbow (Fig.1A) is maintaining a posture or performing a movement, a force is established by the involved muscle based on how much spiking excitation the muscle receives from its $\alpha$-motoneurons (Fig.1B). The $\alpha$-motoneurons are regulated by a variety of sensory input, part of which comes directly from the proprioceptors in the muscle. As the primary proprioceptor found in skeletal muscles, a muscle spindle is another complex system that has its own microscopic Multiple-Input-Multiple-Output structure (Fig.1C). Spindles continuously provide information about the length and lengthening speed of the muscle fiber. A muscle with its regulating motoneurons, sensory neurons and proprioceptors constitutes a monosynaptic spinal loop. This minimalist neurophysiological

structure is used as an example for explaining the multi-scale hyper-time emulation in hardware. Additional structures can be added to the backbone set-up using similar methodologies.

## 2.1 Modularized architecture for multi-scale models

Decades of studies on neurophysiology provided an abundance of models characterizing different components of the human motor nervous system. The informational characteristics of physiological components allowed us to model them as functional structures, i.e. each of which converting input signals to certain outputs. In particular, within a monosynaptic spinal loop illustrated in Fig.1B, stretching the muscle will elicit a chain of physiological activities in: muscle stretch $\Rightarrow$ spindle $\Rightarrow$ sensory neuron $\Rightarrow$ synapse $\Rightarrow$ motoneuron $\Rightarrow$ muscle contraction. The adjacent components must have compatible interfaces, and the interfacing variables must also be physiologically realistic. In our design, each component is mathematically described in Table 1:

Table 1: Functional definition of neural models

| COMPONENT | MATHEMATICAL DEFINITION |
|---|---|
| Neuron | $S(t) = f_{\text{neuron}}(I, t)$ |
| Synapse | $I(t) = f_{\text{synapse}}(S, t)$ |
| Muscle | $T(t) = f_{\text{muscle}}(S, L, \dot{L}, t)$ |
| Spindle | $A(t) = f_{\text{spindle}}(L, \dot{L}, \Gamma_{\text{dynamic}}, \Gamma_{\text{static}}, t)$ |

all components are modeled as black-box functions that map the inputs to the outputs. The meanings of these mathematical definitions are explained below. This design allows existing physiological models to be easily inserted and switched. In all models the input signals are time-varying, e.g. $I = I(t)$, $L = L(t)$, etc. The argument of $t$ in input signals are omitted throughout this paper.

## 2.2 Selection of models for emulation

Models were selected in consideration of their computational cost, physiological verisimilitude, and whether it can be adapted to the mathematical form defined in Table 1.

### Model of Neuron

The informational process for a neuron is to take post-synaptic current $I$ as the input, and produce a binary spike train $S$ in the output. The neuron model adopted in the emulation was developed by Izhikevich [1]:

$$v' = 0.04v^2 + 5v + 140 - u + I \tag{1}$$
$$u' = a(bv - u) \tag{2}$$
$$\text{if } v = 30 \text{ mV, then } v \leftarrow c, \ u \leftarrow u + d$$

where $a, b, c, d$ are free parameters tuned to achieve certain firing patterns. Membrane potential $v$ directly determines a binary spike train $S(t)$ that $S(t) = 1$ if $v \geq 30$, otherwise $S(t) = 0$. Note that $v$ in Izhikevich model is in millivolts and time $t$ is in milliseconds. Therefore the coefficients in eq.1 need to be adjusted in correspondence to SI units.

### Model of Synapse

When a pre-synaptic neuron spikes, i.e. $S(0) = 1$, an excitatory synapse subsequently issues an Excitatory Post-Synaptic Current (EPSC) that drives the post-synaptic neuron. Neural recording of hair cells in rats [2] provided evidence that the time profile of EPSC can be well characterized using the equations below:

$$I(t) = \begin{cases} V_m \times \left( e^{-\frac{t}{\tau_d V_m}} - e^{-\frac{t}{\tau_r V_m}} \right) & \text{if } t \geq 0 \\ 0 & \text{otherwise} \end{cases} \tag{3}$$

The key parameters in a synapse model is the time constants for rising ($\tau_r$) and decaying ($\tau_d$). In our emulation $\tau_r = 0.001$ s and $\tau_r = 0.003$ s.

**Model of Muscle force and electromyograph (EMG)**

The primary effect of skeletal muscle is to convert $\alpha$-motoneuron spikes $S$ into force $T$, depending on the muscle's instantaneous length $L$ and lengthening speed $\dot{L}$. We used Hill's muscle model in the emulation with parameter tuning described in [3]. Another measurable output of muscle is electromyograph (EMG). EMG is the small skin current polarized by motor unit action potential (MUAP) when it travels along muscle fibers. Models exist to describe the typical waveform picked by surface EMG electrodes. In this project we chose to implement the one described in [4].

**Model of Proprioceptor**

Spindle is a sensory organ that provides the main source of proprioceptive information. As can be seen in Fig.1C, a spindle typically produces two afferent outputs (primary Ia and secondary II) according to its gamma fusimotor drives ($\Gamma_{\text{dynamic}}$ and $\Gamma_{\text{static}}$) and muscle states ($L$ and $\dot{L}$). There is currently no closed-form models describing spindle functions due to spindle's significant non-linearity. On representative model that numerically approximates the spindle dynamics was developed by Mileusnic et al. [5]. The model used differential equations to characterize a typical cat soleus spindle. Eqs.4-10 present a subset of this model for one type of spindle fiber (bag1):

$$\dot{x_0} = \left( \frac{\Gamma_{\text{dynamic}}}{\Gamma_{\text{dynamic}} + \Omega_{\text{bag1}}^2} - x_0 \right)/\tau \tag{4}$$

$$\dot{x_1} = x_2 \tag{5}$$

$$\dot{x_2} = \frac{1}{M}\left[ T_{SR} - T_B - T_{PR} - \Gamma_1 x_0 \right] \tag{6}$$

where

$$T_{SR} = K_{SR}(L - x_1 - L_{SR0}) \tag{7}$$

$$T_B = (B_0 + B_1 x_0) \cdot (x_1 - R) \cdot CSS \cdot |x_2|^{0.3} \tag{8}$$

$$T_{PR} = K_{PR}(x_1 - L_{PR0}) \tag{9}$$

$$CSS = \left( \frac{2}{1 + e^{-1000 x_2}} \right) - 1 \tag{10}$$

Eq.8 and 10 suggest that evaluating the spindle model requires multiplication, division as well as more complex arithmetics like polynomials and exponentials. The implementation details are described in Section 3.

## 2.3 Neuron connectivity with sparse interconnections

Although the number of spinal neurons (~1 billion) is significantly less compared to that of cortical neurons (~100 billion), a fully connected spinal network still means approximately 2 trillion synaptic endings [6]. Implementing such a huge number of synapses imposes a major challenge, if not impossible, given limited hardware resource.

In this platform we approximated the neural connectivity by sparsely connecting sensory neurons to motoneurons as parallel pathways. We do not attempt to introduce the full connectivity. The rationale is that in a neural control system, the effect of a single neuron can be considered as mapping current state $x$ to change in state $\dot{x}$ through a band-limited channel. Therefore when a collection of neurons are firing stochastically, the probability of $\dot{x}$ depends on both $x$ and the firing behavior $s$ ($s = 1$ when spiking, otherwise $s = 0$) of each neuron, as such:

$$p(\dot{x}|x, s) = p(\dot{x}|s = 1)p(s = 1|x) + p(\dot{x}|s = 0)p(s = 0|x) \tag{11}$$

Eq.11 is a master equation that determines a probability flow on the state. From the Kramers-Moyal expansion we can associate this probability flow with a partial differential equation:

$$\frac{\partial}{\partial t}p(x, t) = \sum_{i=1}^{\infty} \left( -\frac{\partial}{\partial x} \right)^i D^{(i)}(x)p(x, t) \tag{12}$$

where $D^{(i)}(x)$ is a time-invariant term that modifies the change of probability density based on its $i$-th gradient.

Under certain conditions [7, 8], $D^{(i)}(x)$ for $i > 2$ all vanish and therefore the probability flow can be described deterministically using a linear operator $\mathcal{L}$:

$$\frac{\partial}{\partial t}p(x,t) = \left[-\frac{\partial}{\partial x}D^{(1)}(x) + \frac{\partial^2}{\partial x^2}D^{(2)}(x)\right]p(x,t) = \mathcal{L}p(x,t) \qquad (13)$$

This means that various $\mathcal{L}$s can be superimposed to achieve complex system dynamics (illustrated in Fig.2A).

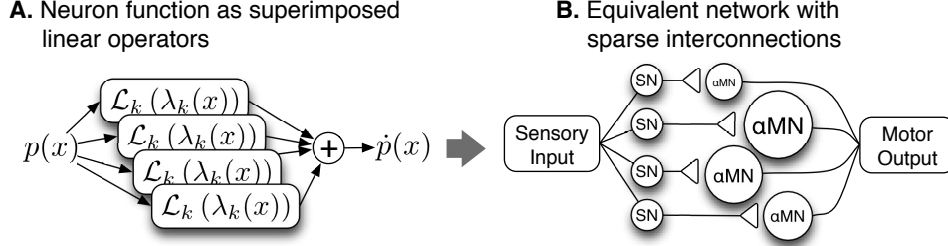

Figure 2: Functions of neuron population can be described as the combination of linear operators (A). Therefore the original neural function can be equivalently produced by sparsely connected neurons formalizing parallel pathways (B).

As a consequence, the statistical effect of two fully connected neuron populations is equivalent to ones that are only sparsely connected, as long as the probability flow can be described by the same $\mathcal{L}$. For a movement task, in particular, it is the statistical effect from the neuron ensemble onto skeletal muscles that determines the global behavior. Therefore we argue that it is feasible to approximate the spinal cord connectivity by sparsely interconnecting sensory and motor neurons (Fig.2B). Here a pool of homogenous sensory neurons projects to another pool of homogeneous $\alpha$-motoneurons. Pseudorandom noise is added to the input of all homogeneous neurons within a population. It is worth noting that this approximation significantly reduces the number of synapses that need to be implemented in hardware.

## 3  Hardware implementation on FPGA

We select FPGA as the implementation device due to its inherent parallelism that resembles the nervous system. FPGA is favored over GPU or clustered CPUs because it is relatively easy to network hundreds of nodes under flexible protocols. The platform is distributed on multiple nodes of Xilinx Spartan-6 devices. The interfacing among FPGAs and computers is created using OpalKelly development board XEM6010. The dynamic range of variables is tight in models of Izhikevich neuron, synapse and EMG. This helps maintaining the accuracy of models even when they are evaluated in 32-bit fixed-point arithmetics. The spindle model, in contrast, requires floating-point arithmetics due to its wide dynamic range and complex calculations (see eq.4-10). Hyper-time computations with floating-point numbers are resource consuming and therefore need to be implemented with special attentions.

### 3.1  Floating-point arithmetics in combinational logic

Our arithmetic implementations are compatible with IEEE-754 standard. Typical floating-point arithmetic IP cores are either pipe-lined or based on iterative algorithms such as CORDIC, all of which require clocks to schedule the calculation. In our platform, no clock is provided for model evaluations thus all arithmetics need to be executed in pure combinational logic. Taking advantage of combinational logic allows all model evaluations to be 1) fast, the evaluation time depends entirely on the propagating and settling time of signals, which is on the order of microseconds, and 2) parallel, each model is evaluated on its own circuit without waiting for any other results.

Our implementations of adder and multiplier are inspired by the open source project "Free Floating-Point Madness", available at http://www.hmc.edu/chips/. Please contact the authors of this paper if the modified code is needed.

**Fast combinational floating-point division**

Floating-point division is even more resource demanding than multiplications. We avoided directly implementing the dividing algorithm by approximating it with additions and multiplications. Our approach is inspired by an algorithm described in [9], which provides a good approximation of the inverse square root for any positive number $x$ within one Newton-Raphson iteration:

$$Q(x) = \frac{1}{\sqrt{x}} \approx x(1.5 - \frac{x}{2} \cdot x^2) \quad (x > 0) \tag{14}$$

$Q(x)$ can be implemented only using floating-point adders and multipliers. Thereby any division with a positive divisor can be achieved if two blocks of $Q(x)$ are concatenated:

$$\frac{a}{b} = \frac{a}{\sqrt{b} \cdot \sqrt{b}} = a \cdot Q(b) \cdot Q(b) \quad (b > 0) \tag{15}$$

This algorithm has been adjusted to also work with negative divisors ($b < 0$).

**Numerical integrators for differential equations**

Evaluating the instantaneous states of differential equation models require a fixed-step numerical integrator. Backward Euler's Method was chosen to balance the numerical error and FPGA usage:

$$\dot{x} = f(x, t) \tag{16}$$
$$x_{n+1} = x_n + Tf(x_{n+1}, t_{n+1}) \tag{17}$$

where $T$ is the sampling interval. $f(x, t)$ is the derivative function for state variable $x$.

### 3.2 Asynchronous spike-based communication between FPGA chips

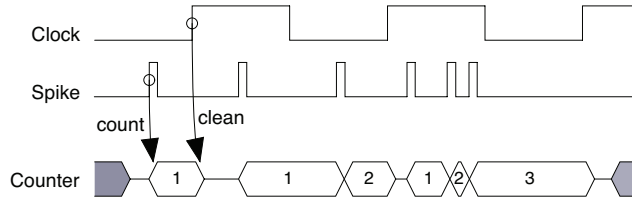

Figure 3: Timing diagram of asynchronous spike-based communication

FPGA nodes are networked by transferring 1-bit binary spikes to each other. Our design allowed the sender and the receiver to operate on independent clocks without having to synchronize. The timing diagram of the spike-based communication is shown in Fig.3. The sender issues Spike with a pulse width of $1/(365 \times F_{\text{emu}})$ second. Each Spike then triggers a counting event on the receiver, meanwhile each Clock first reads the accumulated spike count and subsequently cleans the counter. Note that the phase difference between Spike and Clock is not predictable due to asynchronicity.

### 3.3 Serialize neuron evaluations within a homogeneous population

Different neuron populations are instantiated as standalone circuits. Within in each population, however, homogeneous neurons mentioned in Section 2.3 are evaluated in series in order to optimize FPGA usage.

Within each FPGA node all modules operate with a central clock, which is the only source allowed to trigger any updating event. Therefore the maximal number of neurons that can be serialized ($N_{\text{serial}}$) is restrained by the following relationship:

$$F_{\text{fpga}} = C \times N_{\text{serial}} \times 365 \times F_{\text{emu}} \tag{18}$$

Here $F_{\text{fpga}}$ is the fastest clock rate that a FPGA can operate on; $C = 4$ is the minimal clock cycles needed for updating each state variable in the on-chip memory; $F_{\text{emu}} = 1$ kHz is the time granularity of emulation (1 millisecond), and $365 \times F_{\text{emu}}$ represents 365x real-time. Consider that Xilinx

Spartan-6 FPGA devices peaks at 200MHz central clock frequency, the theoretical maximum of neurons that can be serialized is

$$N_{\mathrm{serial}} \leqslant 200\,\mathrm{MHz}/(4 \times 365 \times 1\,\mathrm{kHz}) \approx 137 \tag{19}$$

In the current design we choose $N_{\mathrm{serial}} = 128$.

# 4 Results: emulated activities of motor nervous system

Figure 4 shows the implemented monosynaptic spinal loop in schematics and in operation. Each FPGA node is able to emulate monosynaptic spinal loops consisting of 1,024 sensory and 1,024 motor neurons, i.e. 2,048 neurons in total. The spike-based asynchronous communication is successful between two FPGA nodes. Note that the emulation has to be significantly slowed down for on-line plotting. When the emulation is at full speed (365x real-time) the software front-end is not able to visualize the signals due to limited data throughput.

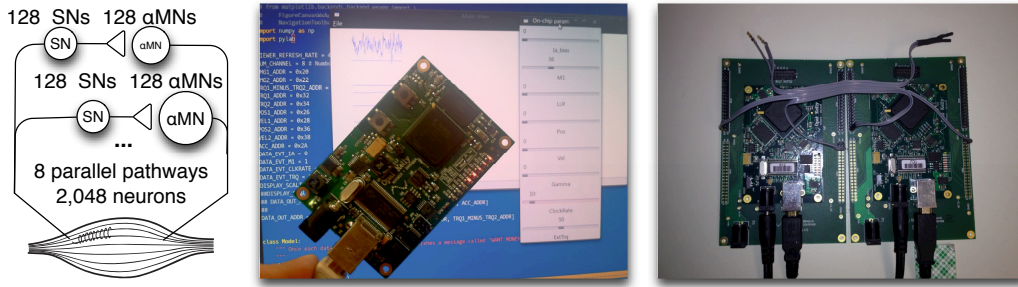

Figure 4: The neural emulation platform in operation. Left: Neural circuits implemented for each FPGA node including 2,048 neurons. SN = Sensory Neuron; $\alpha$MN = $\alpha$-motoneuron. Center: One working FPGA node. Right: Two FPGA nodes networked using asynchronous spiking protocol.

The emulation platform successfully created multi-scale information when the muscle is externally stretched (Fig.5A). We also tested if our emulated motor system is able to produce the recruitment order and size principles observed in real physiological data. It has been well known that when a voluntary motor command is sent to the $\alpha$-motoneuron pool, the motor units are recruited in an order that small ones get recruited first, followed by the big ones [10]. The comparison between our results and real data are shown in Fig.5B, where the top panel shows 20 motor unit activities emulated using our platform, and the bottom panel shows decoded motor unit activities from real human EMG [11]. No qualitative difference was found.

# 5 Discussion and future work

We designed a hardware platform for emulating the multi-scale motor nervous activities in hyper-time. We managed to use one node of single Xilinx Spartan-6 FPGA to emulate monosynaptic spinal loops consisting of 2,048 neurons, associated muscles and proprioceptors. The neurons are organized as parallel pathways with sparse interconnections. The emulation is successfully accelerated to 365x real-time. The platform can be scaled by networking multiple FPGA nodes, which is enabled by an asynchronous spike-based communication protocol. The emulated monosynaptic spinal loops are capable of producing reflex-like activities in response to muscle stretch. Our results of motor unit recruitment order are compatible with the physiological data collected in real human subjects.

There is a question of whether this stochastic system turns out chaotic, especially with accumulated errors from Backward Euler's integrator. Note that the firing property of a neuron population is usually stable even with explicit noise [8], and spindle inputs are measured from real robots so the integrator errors are corrected at every iteration. To our knowledge, the system is not critically sensitive to the initial conditions or integrator errors. This question, however, is both interesting and important for in-depth investigations in the future.

It has been shown [12] that replicating classic types of spinal interneurons (propriospinal, Ia-excitatory, Ia-inhibitory, Renshaw, etc.) is sufficient to produce stabilizing responses and rapid reaching movement in a wrist. Our platform will introduce those interneurons to describe the known spinal circuitry in further details. Physiological models will also be refined as needed. For the purpose of modeling movement behavior or diseases, Izhikevich model is a good balance between verisimilitude and computational cost. Nevertheless when testing drug effects along disease progression, neuron models are expected to cover sufficient molecular details including how neurotransmitters affect various ion channels. With the advancing of programmable semiconductor technology, it is expected to upgrade our neuron model to Hodgkin-Huxley's. For the muscle models, Hill's type of model does not fit the muscle properties accurately enough when the muscle is being shortened. Alternative models will be tested.

Other studies showed that the functional dexterity of human limbs – especially in the hands – is critically enabled by the tendon configurations and joint geometry [13]. As a result, if our platform is used to understand whether known neurophysiology and biomechanics are sufficient to produce able and pathological movements, it will be necessary to use this platform to control human-like limbs. Since the emulation speed can be flexibly adjusted from arbitrarily slow to 365x real-time, when speeded to exactly 1x real-time the platform will function as a digital controller with 1kHz refresh rate.

The main purpose of the emulation is to learn how certain motor disorders progress during childhood development. This first requires the platform to reproduce motor symptoms that are compatible with clinical observations. For example it has been suggested that muscle spasticity in rats is associated with decreased soma size of $\alpha$-motoneurons [14], which presumably reduced the firing threshold of neurons. Thus when lower firing threshold is introduced to the emulated motoneuron pool, similar EMG patterns as in [15] should be observed. It is also necessary for the symptoms to evolve with neural plasticity. In the current version we presume that the structure of each component remains time invariant. In the future work Spike Timing Dependent Plasticity (STDP) will be introduced such that all components are subject to temporal modifications.

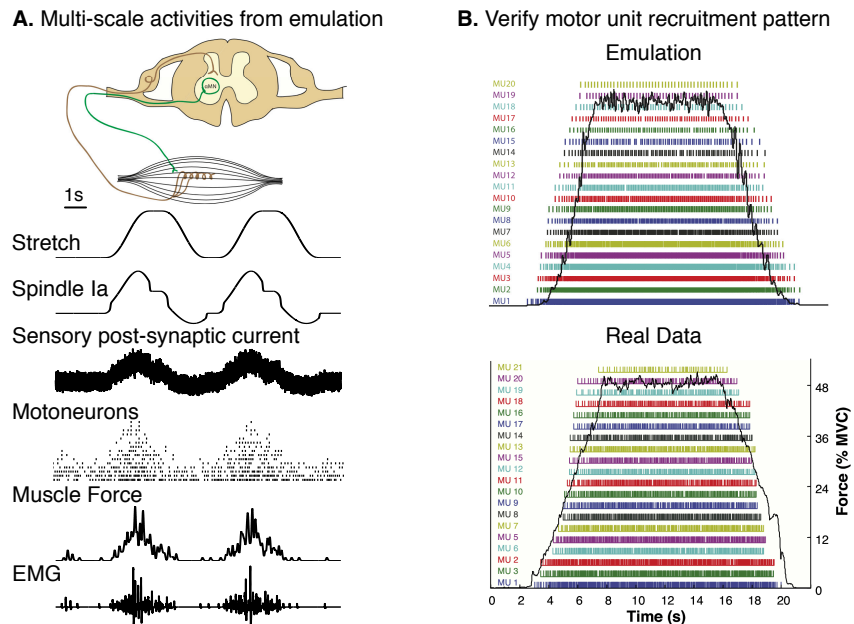

Figure 5: A) Physiological activity emulated by each model when the muscle is sinusoidally stretched. B) Comparing the emulated motor unit recruitment order with real experimental data.

## Acknowledgments

The authors thank Dr. Gerald Loeb for helping set up the emulation of spindle models. This project is supported by NIH NINDS grant R01NS069214-02.

## References

[1] Izhikevich, E. M. Simple model of spiking neurons. *IEEE transactions on neural networks / a publication of the IEEE Neural Networks Council* **14**, 1569–1572 (2003).

[2] Glowatzki, E. & Fuchs, P. A. Transmitter release at the hair cell ribbon synapse. *Nature neuroscience* **5**, 147–154 (2002).

[3] Shadmehr, R. & Wise, S. P. A Mathematical Muscle Model. In *Supplementary documents for "Computational Neurobiology of Reaching and Pointing"*, 1–18 (MIT Press, Cambridge, MA, 2005).

[4] Fuglevand, A. J., Winter, D. A. & Patla, A. E. Models of recruitment and rate coding organization in motor-unit pools. *Journal of neurophysiology* **70**, 2470–2488 (1993).

[5] Mileusnic, M. P., Brown, I. E., Lan, N. & Loeb, G. E. Mathematical models of proprioceptors. I. Control and transduction in the muscle spindle. *Journal of neurophysiology* **96**, 1772–1788 (2006).

[6] Gelfan, S., Kao, G. & Ruchkin, D. S. The dendritic tree of spinal neurons. *The Journal of comparative neurology* **139**, 385–411 (1970).

[7] Sanger, T. D. Neuro-mechanical control using differential stochastic operators. In *Engineering in Medicine and Biology Society (EMBC), 2010 Annual International Conference of the IEEE*, 4494–4497 (2010).

[8] Sanger, T. D. Distributed control of uncertain systems using superpositions of linear operators. *Neural computation* **23**, 1911–1934 (2011).

[9] Lomont, C. Fast inverse square root (2003). URL http://www.lomont.org/Math/Papers/2003/InvSqrt.pdf.

[10] Henneman, E. Relation between size of neurons and their susceptibility to discharge. *Science (New York, N.Y.)* **126**, 1345–1347 (1957).

[11] De Luca, C. J. & Hostage, E. C. Relationship between firing rate and recruitment threshold of motoneurons in voluntary isometric contractions. *Journal of neurophysiology* **104**, 1034–1046 (2010).

[12] Raphael, G., Tsianos, G. A. & Loeb, G. E. Spinal-like regulator facilitates control of a two-degree-of-freedom wrist. *The Journal of neuroscience : the official journal of the Society for Neuroscience* **30**, 9431–9444 (2010).

[13] Valero-Cuevas, F. J. *et al.* The tendon network of the fingers performs anatomical computation at a macroscopic scale. *IEEE transactions on bio-medical engineering* **54**, 1161–1166 (2007).

[14] Brashear, A. & Elovic, E. *Spasticity: Diagnosis and Management* (Demos Medical, 2010), 1 edn.

[15] Levin, M. F. & Feldman, A. G. The role of stretch reflex threshold regulation in normal and impaired motor control. *Brain research* **657**, 23–30 (1994).

